# RTRMC: A Riemannian trust-region method for low-rank matrix completion

**Nicolas Boumal**[*]
ICTEAM Institute
Université catholique de Louvain
B-1348 Louvain-la-Neuve
nicolas.boumal@uclouvain.be

**P.-A. Absil**
ICTEAM Institute
Université catholique de Louvain
B-1348 Louvain-la-Neuve
absil@inma.ucl.ac.be

## Abstract

We consider large matrices of low rank. We address the problem of recovering such matrices when most of the entries are unknown. Matrix completion finds applications in recommender systems. In this setting, the rows of the matrix may correspond to items and the columns may correspond to users. The known entries are the ratings given by users to some items. The aim is to predict the unobserved ratings. This problem is commonly stated in a constrained optimization framework. We follow an approach that exploits the geometry of the low-rank constraint to recast the problem as an unconstrained optimization problem on the Grassmann manifold. We then apply first- and second-order Riemannian trust-region methods to solve it. The cost of each iteration is linear in the number of known entries. Our methods, RTRMC 1 and 2, outperform state-of-the-art algorithms on a wide range of problem instances.

## 1 Introduction

We address the problem of recovering a low-rank $m$-by-$n$ matrix $X$ of which a few entries are observed, possibly with noise. Throughout, we assume that $r = \mathrm{rank}(X) \ll m \le n$ and note $\Omega \subset \{1 \ldots m\} \times \{1 \ldots n\}$ the set of indices of the observed entries of $X$, i.e., $X_{ij}$ is known iff $(i, j) \in \Omega$. Solving this problem is namely useful in recommender systems, where one tries to predict the ratings users would give to items they have not purchased.

### 1.1 Related work

In the noiseless case, one could state the minimum rank matrix recovery problem as follows:

$$\min_{\hat{X} \in \mathbb{R}^{m \times n}} \mathrm{rank}\, \hat{X}, \text{ such that } \hat{X}_{ij} = X_{ij}\ \forall (i, j) \in \Omega. \tag{1}$$

This problem, however, is NP hard [CR09]. A possible convex relaxation of (1) introduced by Candès and Recht [CR09] is to use the nuclear norm of $\hat{X}$ as objective function, i.e., the sum of its singular values, noted $\|\hat{X}\|_*$. The SVT method [CCS08] attempts to solve such a convex problem using tools from compressed sensing and the ADMiRA method [LB10] does so using matching pursuit-like techniques.

Alternatively, one may minimize the discrepancy between $\hat{X}$ and $X$ at entries $\Omega$ under the constraint that $\mathrm{rank}(\hat{X}) \le r$ for some small constant $r$. Since any matrix $\hat{X}$ of rank at most $r$ may be written in the form $UW$ with $U \in \mathbb{R}^{m \times r}$ and $W \in \mathbb{R}^{r \times n}$, a reasonable formulation of the problem reads:

$$\min_{U \in \mathbb{R}^{m \times r}} \min_{W \in \mathbb{R}^{r \times n}} \sum_{(i,j) \in \Omega} \left((UW)_{ij} - X_{ij}\right)^2. \tag{2}$$

---

[*]Web: http://perso.uclouvain.be/nicolas.boumal/

The LMaFit method [WYZ10] does a good job at solving this problem by alternatively fixing either of the variables and solving the resulting least-squares problem efficiently.

One drawback of the latter formulation is that the factorization of a matrix $\hat{X}$ into the product $UW$ is not unique. Indeed, for any $r$-by-$r$ invertible matrix $M$, we have $UW = (UM)(M^{-1}W)$. All the matrices $UM$ share the same column space. Hence, the optimal value of the inner optimization problem in (2) is a function of $\text{col}(U)$—the column space of $U$—rather than $U$ specifically. Dai et al. [DMK11, DKM10] exploit this to recast (2) on the Grassmann manifold $\mathcal{G}(m, r)$, i.e., the set of $r$-dimensional vector subspaces of $\mathbb{R}^m$ (see Section 2):

$$\min_{\mathscr{U} \in \mathcal{G}(m,r)} \min_{W \in \mathbb{R}^{r \times n}} \sum_{(i,j) \in \Omega} \big((UW)_{ij} - X_{ij}\big)^2, \tag{3}$$

where $U \in \mathbb{R}^{m \times r}$ is any matrix such that $\text{col}(U) = \mathscr{U}$ and is often chosen to be orthonormal. Unfortunately, the objective function of the outer minimization in (3) may be discontinuous at points $\mathscr{U}$ for which the least-squares problem in $W$ does not have a unique solution. Dai et al. proposed ingenious ways to deal with the discontinuity. Their focus, though, was on deriving theoretical performance guarantees rather than developing fast algorithms.

Keshavan et al. [KO09, KM10] state the problem on the Grassmannian too, but propose to simultaneously optimize on the row and column spaces, yielding a smaller least-squares problem which is unlikely to not have a unique solution, resulting in a smooth objective function. In one of their recent papers [KM10], they solve:

$$\min_{\mathscr{U} \in \mathcal{G}(m,r), \mathscr{V} \in \mathcal{G}(n,r)} \min_{S \in \mathbb{R}^{r \times r}} \sum_{(i,j) \in \Omega} \big((USV^\top)_{ij} - X_{ij}\big)^2 + \lambda^2 \left\| USV^\top \right\|_{\text{F}}^2, \tag{4}$$

where $U$ and $V$ are any orthonormal bases of $\mathscr{U}$ and $\mathscr{V}$, respectively, and $\lambda$ is a regularization parameter. The authors propose an efficient SVD-based initial guess for $\mathscr{U}$ and $\mathscr{V}$ which they refine using a steepest descent method, along with strong theoretical guarantees.

Meyer et al. [MBS11] proposed a Riemannian approach to linear regression on fixed-rank matrices. Their regression framework encompasses matrix completion problems. Likewise, Balzano et al. [BNR10] introduced GROUSE for subspace identification on the Grassmannian, applicable to matrix completion. Finally, in the preprint [Van11] which became public while we were preparing the camera-ready version of this paper, Vandereycken proposes an approach based on the submanifold geometry of the sets of fixed-rank matrices.

## 1.2 Our contribution and outline of the paper

Dai et al.'s initial formulation (3) has a discontinuous objective function on the Grassmannian. The OptSpace formulation (4) on the other hand has a continuous objective and comes with a smart initial guess, but optimizes on a higher-dimensional search space, while it is arguably preferable to keep the dimension of the manifold search space low, even at the expense of a larger least-squares problem. Furthermore, the OptSpace regularization term is efficiently computable since $\left\| USV^\top \right\|_{\text{F}} = \| S \|_{\text{F}}$, but it penalizes all entries instead of just the entries $(i, j) \notin \Omega$.

In an effort to combine the best of both worlds, we equip (3) with a regularization term weighted by $\lambda > 0$, which yields a smooth objective function defined over an appropriate search space:

$$\min_{\mathscr{U} \in \mathcal{G}(m,r)} \min_{W \in \mathbb{R}^{r \times n}} \frac{1}{2} \sum_{(i,j) \in \Omega} C_{ij}^2 \big((UW)_{ij} - X_{ij}\big)^2 + \frac{\lambda^2}{2} \sum_{(i,j) \notin \Omega} (UW)_{ij}^2. \tag{5}$$

Here, we introduced a confidence index $C_{ij} > 0$ for each observation $X_{ij}$, which may be useful in applications. As we will see, introducing a regularization term is essential to ensure smoothness of the objective and hence obtain good convergence properties. It may not be critical for practical problem instances though.

We further innovate on previous works by using a Riemannian trust-region method, GenRTR [ABG07], as optimization algorithm to minimize (5) on the Grassmannian. GenRTR is readily available as a free Matlab package and comes with strong convergence results that are naturally inherited by our algorithms.

In Section 2, we rapidly cover the essential useful tools on the Grassmann manifold. In Section 3, we derive expressions for the gradient and the Hessian of our objective function while paying special attention to complexity. Section 4 sums up the main properties of the Riemannian trust-region method. Section 5 shows a few results of numerical experiments demonstrating the effectiveness of our approach.

## 2 Geometry of the Grassmann manifold

Our objective function $f$ (10) is defined over the Grassmann manifold $\mathcal{G}(m,r)$, i.e., the set of $r$-dimensional vector subspaces of $\mathbb{R}^m$. Absil et al. [AMS08] give a computation-oriented description of the geometry of this manifold. Here, we only give a summary of the important tools we use.

Each point $\mathscr{U} \in \mathcal{G}(m,r)$ is a vector subspace we may represent numerically as the column space of a full-rank matrix $U \in \mathbb{R}^{m \times r}$: $\mathscr{U} = \mathrm{col}(U)$. For numerical reasons, we will only use orthonormal matrices $U \in \mathcal{U}(m,r) = \{U \in \mathbb{R}^{m \times r} : U^\top U = I_r\}$. The set $\mathcal{U}(m,r)$ is the Stiefel manifold.

The Grassmannian is a Riemannian manifold, and as such we can define a tangent space to $\mathcal{G}(m,r)$ at each point $\mathscr{U}$, noted $\mathrm{T}_{\mathscr{U}}\mathcal{G}(m,r)$. The latter is a vector space of dimension $\dim \mathcal{G}(m,r) = r(m-r)$. A tangent vector $\mathscr{H} \in \mathrm{T}_{\mathscr{U}}\mathcal{G}(m,r)$, where we represent $\mathscr{U}$ as the orthonormal matrix $U$, is represented by a unique matrix $H \in \mathbb{R}^{m \times r}$ verifying $U^\top H = 0$ and $\frac{\mathrm{d}}{\mathrm{d}t}\mathrm{col}(U+tH)\big|_{t=0} = \mathscr{H}$. For practical purposes we may, with a slight abuse of notation we often commit hereafter, write $\mathscr{H} = H$—assuming $U$ is known from the context—and $\mathrm{T}_U\mathcal{G}(m,r) = \{H \in \mathbb{R}^{m \times r} : U^\top H = 0\}$. Each tangent space is endowed with an inner product, the Riemannian metric, that varies smoothly from point to point. It is inherited from the embedding space of the matrix representation of tangent vectors $\mathbb{R}^{m \times r}$: $\forall H_1, H_2 \in \mathrm{T}_U\mathcal{G}(m,r) : \langle H_1, H_2 \rangle_U = \mathrm{Trace}(H_2^\top H_1)$. The orthogonal projector from $\mathbb{R}^{m \times r}$ onto the tangent space $\mathrm{T}_U\mathcal{G}(m,r)$ is given by:

$$P_U : \mathbb{R}^{m \times r} \to \mathrm{T}_U\mathcal{G}(m,r) : H \mapsto P_U H = (I - UU^\top)H.$$

One can also project a vector onto the tangent space of the Stiefel manifold:

$$P_U^{\mathrm{St}} : \mathbb{R}^{m \times r} \to \mathrm{T}_U\mathcal{U}(m,r) : H \mapsto P_U^{\mathrm{St}} H = (I - UU^\top)H + U\mathrm{skew}(U^\top H),$$

where $\mathrm{skew}(X) = (X - X^\top)/2$ extracts the skew-symmetric part of $X$. This is useful for the computation of $\mathrm{grad}f(U) \in \mathrm{T}_U\mathcal{G}(m,r)$. Indeed, according to [AMS08, eqs. (3.37) and (3.39)], considering $\bar{f} : \mathbb{R}^{m \times r} \to \mathbb{R}$, its restriction $\bar{f}\big|_{\mathcal{U}(m,r)}$ to the Stiefel manifold and $f : \mathcal{G}(m,r) \to \mathbb{R}$ such that $f(\mathrm{col}(U)) = \bar{f}\big|_{\mathcal{U}(m,r)}(U)$ is well-defined, as will be the case in Section 3, we have (with a slight abuse of notation):

$$\mathrm{grad}f(U) = \mathrm{grad}\,\bar{f}\big|_{\mathcal{U}(m,r)}(U) = P_U^{\mathrm{St}}\mathrm{grad}\bar{f}(U). \tag{6}$$

Similarly, since $P_U \circ P_U^{\mathrm{St}} = P_U$, the Hessian of $f$ at $\mathscr{U}$ along $\mathscr{H}$ is given by [AMS08, eqs. (5.14) and (5.18)]:

$$\mathrm{Hess}f(U)[H] = P_U(\mathrm{D}(U \mapsto P_U^{\mathrm{St}}\mathrm{grad}\bar{f}(U))(U)[H]), \tag{7}$$

where $\mathrm{D}g(X)[H]$ is the directional derivative of $g$ at $X$ along $H$, in the classical sense. For our optimization algorithms, it is important to be able to move along the manifold from some initial point $U$ in some prescribed direction specified by a tangent vector $H$. To this end, we use the retraction:

$$\mathrm{R}_U(H) = \mathrm{qf}(U + H), \tag{8}$$

where $\mathrm{qf}(X) \in \mathcal{U}(m,r)$ designates the $m$-by-$r$ $Q$-factor of the $QR$ decomposition of $X \in \mathbb{R}^{m \times r}$.

## 3 Computation of the objective function and its derivatives

We seek an $m$-by-$n$ matrix $\hat{X}$ of rank not more than $r$ such that $\hat{X}$ is as close as possible to a given matrix $X$ at the entries in the observation set $\Omega$. Furthermore, we are given a weight matrix $C \in \mathbb{R}^{m \times n}$ indicating the confidence we have in each observed entry of $X$. The matrix $C$ is positive at entries in $\Omega$ and zero elsewhere. To this end, we consider the following function, where $(X_\Omega)_{ij}$ equals $X_{ij}$ if $(i,j) \in \Omega$ and is zero otherwise:

$$\hat{f} : \mathbb{R}^{m \times r} \times \mathbb{R}^{r \times n} \to \mathbb{R} : (U, W) \mapsto \hat{f}(U, W) = \frac{1}{2}\|C \odot (UW - X_\Omega)\|_\Omega^2 + \frac{\lambda^2}{2}\|UW\|_{\bar{\Omega}}^2, \tag{9}$$

where $\odot$ is the entry-wise product, $\lambda > 0$ is a regularization parameter, $\bar{\Omega}$ is the complement of the set $\Omega$ and $\|M\|_\Omega^2 \triangleq \sum_{(i,j) \in \Omega} M_{ij}^2$. Picking a small but positive $\lambda$ will ensure that the objective function $f$ (10) is smooth. For a fixed $U$, computing the matrix $W$ that minimizes $\hat{f}$ is a least-squares problem. The mapping between $U$ and this (unique) optimal $W$, noted $W_U$,

$$U \mapsto W_U = \underset{W \in \mathbb{R}^{r \times n}}{\mathrm{argmin}}\,\hat{f}(U, W),$$

is smooth and easily computable—see Section 3.3.

By virtue of the discussion in Section 1, we know that the mapping $U \mapsto \hat{f}(U, W_U)$, with $U \in \mathbb{R}^{m \times r}$, is constant over sets of full-rank matrices $U$ spanning the same column space. Hence, considering these sets as equivalence classes $\mathscr{U}$, the following function $f$ over the Grassmann manifold is well-defined:

$$f : \mathcal{G}(m, r) \to \mathbb{R} : \mathscr{U} \mapsto f(\mathscr{U}) = \hat{f}(U, W_U), \tag{10}$$

with any full-rank $U \in \mathbb{R}^{m \times r}$ such that $\mathrm{col}(U) = \mathscr{U}$. The interpretation is as follows: we are looking for an optimal matrix $\hat{X} = UW$ of rank at most $r$; we have confidence $C_{ij}$ that $\hat{X}_{ij}$ should equal $X_{ij}$ for $(i, j) \in \Omega$ and (very small) confidence $\lambda$ that $\hat{X}_{ij}$ should equal 0 for $(i, j) \notin \Omega$.

## 3.1 Rearranging the objective

Considering (9), it looks like evaluating $\hat{f}(U, W)$ will require the computation of the product $UW$ at the entries in $\Omega$ *and* $\bar{\Omega}$, i.e., we would need to compute the whole matrix $UW$, which cannot cost much less than $\mathcal{O}(mnr)$. Since applications typically involve very large values of the product $mn$, this is not acceptable. Alternatively, if we restrict ourselves—without loss of generality—to orthonormal matrices $U$, we observe that

$$\|UW\|_\Omega^2 + \|UW\|_{\bar{\Omega}}^2 = \|UW\|_\mathrm{F}^2 = \|W\|_\mathrm{F}^2.$$

Consequently, for all $U$ in $\mathcal{U}(m, r)$, we have $\hat{f}(U, W_U) = \check{f}(U, W_U)$, where

$$\check{f}(U, W) = \frac{1}{2} \|C \odot (UW - X_\Omega)\|_\Omega^2 + \frac{\lambda^2}{2} \|W\|_\mathrm{F}^2 - \frac{\lambda^2}{2} \|UW\|_\Omega^2. \tag{11}$$

This only requires the computation of $UW$ at entries in $\Omega$, at a cost of $\mathcal{O}(|\Omega|r)$. Finally, let $\bar{f} : \mathbb{R}^{m \times r} \to \mathbb{R} : U \mapsto \check{f}(U, W_U)$, and observe that $f(\mathrm{col}(U)) = \bar{f}|_{\mathcal{U}(m,r)}(U)$ for all $U$ in $\mathcal{U}(m, r)$, as in the setting of Section 2.

## 3.2 Gradient and Hessian of the objective

We now derive formulas for the first and second order derivatives of $f$. In deriving these formulas, it is useful to note that, for a suitably smooth mapping $g$,

$$\mathrm{grad}\big(X \mapsto 1/2 \|g(X)\|_\mathrm{F}^2\big)(X) = \big(\mathrm{D}g(X)\big)^*[g(X)], \tag{12}$$

where $\big(\mathrm{D}g(X)\big)^*$ is the adjoint of the differential of $g$ at $X$. For ease of notation, let us define the following $m$-by-$n$ matrix with the sparsity structure induced by $\Omega$:

$$\hat{C}_{ij} = \begin{cases} C_{ij}^2 - \lambda^2 & \text{if } (i, j) \in \Omega, \\ 0 & \text{otherwise.} \end{cases} \tag{13}$$

We also introduce a sparse residue matrix $R_U$ that will come up in various formulas:

$$R_U = \hat{C} \odot (UW_U - X_\Omega) - \lambda^2 X_\Omega. \tag{14}$$

Successively using the chain rule, the optimality of $W_U$ and (12), we obtain:

$$\mathrm{grad}\bar{f}(U) = \frac{\mathrm{d}}{\mathrm{d}U} \check{f}(U, W_U) = \frac{\partial}{\partial U} \check{f}(U, W_U) + \frac{\partial}{\partial W} \check{f}(U, W_U) \cdot \frac{\mathrm{d}}{\mathrm{d}U} W_U = \frac{\partial}{\partial U} \check{f}(U, W_U) = R_U W_U^\top.$$

Indeed, since $W_U$ is optimal, $\frac{\partial}{\partial W} \check{f}(U, W_U) = U^\top R_U + \lambda^2 W_U = 0$. Then, according to the identity (6) and since $U^\top R_U = -\lambda^2 W_U$, the gradient of $f$ at $\mathscr{U} = \mathrm{col}(U)$ on the Grassmannian is given by:

$$\mathrm{grad}f(U) = \mathrm{grad}\ \bar{f}|_{\mathcal{U}(m,r)}(U) = P_U^\mathrm{St} \mathrm{grad}\bar{f}(U) = (I - UU^\top)R_U W_U^\top + U\mathrm{skew}(U^\top R_U W_U^\top)$$

$$= (I - UU^\top)R_U W_U^\top - \lambda^2 U\mathrm{skew}(W_U W_U^\top) = R_U W_U^\top + \lambda^2 U(W_U W_U^\top), \tag{15}$$

We now differentiate (15) according to the identity (7) to get a matrix representation of the Hessian of $f$ at $\mathscr{U}$ along $\mathscr{H}$. We note $H$ a matrix representation of the tangent vector $\mathscr{H}$ chosen in accordance with $U$ and

$$W_{U,H} \triangleq \mathrm{D}(U \mapsto W_U)(U)[H]$$

the derivative of the mapping $U \mapsto W_U$ at $U$ along the tangent direction $H$. Then:

$$\mathrm{Hess}f(U)[H] = (I - UU^\top)\mathrm{Dgrad}f(U)[H]$$

$$= (I - UU^\top)\big[\hat{C} \odot (HW_U + UW_{U,H})\big]W_U^\top + R_U W_{U,H}^\top + \lambda^2 H(W_U W_U^\top) + \lambda^2 U(W_U W_{U,H}^\top). \tag{16}$$

### 3.3 $W_U$ and its derivative $W_{U,H}$

We still need to provide an explicit formula for $W_U$ and $W_{U,H}$. We assume $U \in \mathcal{U}(m,r)$ since we use orthonormal matrices to represent points on the Grassmannian and $U^\top H = 0$ since $\mathscr{H}$ is a tangent vector at $\mathscr{U}$.

We use the vectorization operator, vec, that transforms matrices into vectors by stacking their columns—in Matlab notation, $\mathrm{vec}(A) = A(:)$. Denoting the Kronecker product of two matrices by $\otimes$, we will use the well-known identity for matrices $A, Y, B$ of appropriate sizes [Bro05]:

$$\mathrm{vec}(AYB) = (B^\top \otimes A)\mathrm{vec}(Y).$$

We also write $I_\Omega$ for the orthonormal $|\Omega|$-by-$mn$ matrix such that $\mathrm{vec}_\Omega(M) = I_\Omega \mathrm{vec}(M)$ is a vector of length $|\Omega|$ corresponding to the entries $M_{ij}$ for $(i,j) \in \Omega$, taken in order from $\mathrm{vec}(M)$.

Computing $W_U$ comes down to minimizing the least-squares objective $\check{f}(U,W)$ (11) with respect to $W$. We first manipulate $\check{f}$ to reach a standard form for least-squares, with $S = I_\Omega \mathrm{diag}(\mathrm{vec}(C))$:

$$\begin{aligned}
\check{f}(U,W) &= \frac{1}{2} \|C \odot (UW - X_\Omega)\|_\Omega^2 + \frac{\lambda^2}{2} \|W\|_F^2 - \frac{\lambda^2}{2} \|UW\|_\Omega^2 \\
&= \frac{1}{2} \|S\mathrm{vec}(UW) - \mathrm{vec}_\Omega(C \odot X_\Omega)\|_2^2 + \frac{\lambda^2}{2} \|\mathrm{vec}(W)\|_2^2 - \frac{\lambda^2}{2} \|\mathrm{vec}_\Omega(UW)\|_2^2 \\
&= \frac{1}{2} \|S(I_n \otimes U)\mathrm{vec}(W) - \mathrm{vec}_\Omega(C \odot X_\Omega)\|_2^2 + \frac{1}{2} \|\lambda I_{rn}\mathrm{vec}(W)\|_2^2 - \frac{1}{2} \|\lambda I_\Omega(I_n \otimes U)\mathrm{vec}(W)\|_2^2 \\
&= \frac{1}{2} \left\| \begin{bmatrix} S(I_n \otimes U) \\ \lambda I_{rn} \end{bmatrix} \mathrm{vec}(W) - \begin{bmatrix} \mathrm{vec}_\Omega(C \odot X_\Omega) \\ 0_{rn} \end{bmatrix} \right\|_2^2 - \frac{1}{2} \|[\lambda I_\Omega(I_n \otimes U)]\mathrm{vec}(W)\|_2^2 \\
&= \frac{1}{2} \|A_1 w - b_1\|_2^2 - \frac{1}{2} \|A_2 w\|_2^2,
\end{aligned}$$

where $w = \mathrm{vec}(W) \in \mathbb{R}^{rn}$, $0_{rn} \in \mathbb{R}^{rn}$ is the zero-vector and the definitions for $A_1, A_2$ and $b_1$ are obvious. If $A_1^\top A_1 - A_2^\top A_2$ is positive definite there is a unique minimizing vector $\mathrm{vec}(W_U)$, given by:

$$\mathrm{vec}(W_U) = (A_1^\top A_1 - A_2^\top A_2)^{-1} A_1^\top b_1.$$

It is easy to compute the following:

$$\begin{aligned}
A_1^\top A_1 &= (I_n \otimes U^\top)(S^\top S)(I_n \otimes U) + \lambda^2 I_{rn}, \\
A_2^\top A_2 &= (I_n \otimes U^\top)(\lambda^2 I_\Omega^\top I_\Omega)(I_n \otimes U), \\
A_1^\top b_1 &= (I_n \otimes U^\top)S^\top \mathrm{vec}_\Omega(C \odot X_\Omega) = (I_n \otimes U^\top)\mathrm{vec}(C^{(2)} \odot X_\Omega).
\end{aligned}$$

Throughout the text, we use the notation $M^{(n)}$ for entry-wise exponentiation, i.e., $(M^{(n)})_{ij} = (M_{ij})^n$. Note that $S^\top S - \lambda^2 I_\Omega^\top I_\Omega = \mathrm{diag}(\mathrm{vec}(\hat{C}))$. We then define $A \in \mathbb{R}^{rn \times rn}$ as:

$$A \triangleq A_1^\top A_1 - A_2^\top A_2 = (I_n \otimes U^\top)\left(\mathrm{diag}(\mathrm{vec}(\hat{C}))\right)(I_n \otimes U) + \lambda^2 I_{rn}. \tag{17}$$

Observe that the matrix $A$ is block-diagonal, with $n$ symmetric blocks of size $r$. Each block is indeed positive-definite provided $\lambda > 0$ (making $A$ positive-definite too). Thanks to the sparsity of $\hat{C}$, we can compute these $n$ blocks with $\mathcal{O}(|\Omega|r^2)$ flops. To solve systems in $A$, we compute the Cholesky factorization of each block, at a total cost of $\mathcal{O}(nr^3)$. Once these factorizations are computed, each system only costs $\mathcal{O}(nr^2)$ to solve [TB97].

Collecting all equations in this subsection, we obtain a closed-form formula for $W_U$:

$$\mathrm{vec}(W_U) = A^{-1}\mathrm{vec}\left(U^\top[C^{(2)} \odot X_\Omega]\right), \tag{18}$$

where $A$ is a function of $U$. We would like to differentiate $W_U$ with respect to $U$. Using bilinearity and associativity of $\otimes$ as well as the formula $\mathrm{D}(Y \mapsto Y^{-1})(X)[H] = -X^{-1}HX^{-1}$ [Bro05], some algebra yields:

$$\mathrm{vec}(W_{U,H}) = -A^{-1}\mathrm{vec}\left(H^\top R_U + U^\top(\hat{C} \odot (HW_U))\right). \tag{19}$$

The most expensive operation involved in computing $W_{U,H}$ ought to be the resolution of a linear system in $A$. Fortunately, we already factored the $n$ small diagonal blocks of $A$ in Cholesky form to compute $W_U$. Consequently, after computing $W_U$, computing $W_{U,H}$ is cheaper than computing $W_{U'}$ for a new $U'$. This means that we can benefit from computing this information before we move on to a new candidate on the Grassmannian, i.e., it is worth trying second order methods. We summarize the complexities in the next subsection.

### 3.4 Numerical complexities

By exploiting the sparsity of many of the matrices involved and the special structure of the matrix $A$ appearing in the computation of $W_U$ and $W_{U,H}$, it is possible to compute the objective $f$ as well as its gradient and its Hessian on the Grassmannian in time essentially linear in the size of the data $|\Omega|$. Memory complexities are also linear in $|\Omega|$. We summarize the computational complexities in Table 1. Please note that most computations are easily parallelizable, but we do not take advantage of it here.

Table 1: All complexities are essentially linear in $|\Omega|$, the number of observed entries.

| Computation | Complexity | By-products | Formulas |
|---|---|---|---|
| $W_U$ and $f(U)$ | $\mathcal{O}(|\Omega|r^2 + nr^3)$ | Cholesky form of $A$ | (9), (10), (17), (18) |
| $\mathrm{grad}f(U)$ | $\mathcal{O}(|\Omega|r + (m+n)r^2)$ | $R_U$ and $W_U W_U^\top$ | (13), (14), (15) |
| $W_{U,H}$ and $\mathrm{Hess}f(U)[H]$ | $\mathcal{O}(|\Omega|r + (m+n)r^2)$ | | (16), (19) |

## 4  Riemannian trust-region method

We use a Riemannian trust-region (RTR) method [ABG07] to minimize (10), via the freely available Matlab package GenRTR (version 0.3.0) with its default parameter values. The package is available at this address: `http://www.math.fsu.edu/~cbaker/GenRTR/?page=download`.

At the current iterate $\mathscr{U} = \mathrm{col}(U)$, the RTR method uses the retraction $\mathrm{R}_U$ (8) to build a quadratic model $m_U : \mathrm{T}_U\mathcal{G}(m,r) \to \mathbb{R}$ of the lifted objective function $f \circ \mathrm{R}_U$ (*lift*). It then classically minimizes the model inside a trust region on this vector space (*solve*), and retracts the resulting tangent vector $H$ to a candidate $U^+ = \mathrm{R}_U(H)$ on the Grassmannian (*retract*). The quality of $\mathscr{U}^+ = \mathrm{col}(U^+)$ is assessed using $f$ and the step is accepted or rejected accordingly. Likewise, the radius of the trust region is adapted based on the observed quality of the model.

The model $m_U$ of $f \circ \mathrm{R}_U$ has the form:

$$m_U(H) = f(U) + \langle \mathrm{grad}f(U), H \rangle_U + \frac{1}{2}\langle A(U)[H], H \rangle_U,$$

where $A(U)$ is some symmetric linear operator on $\mathrm{T}_U\mathcal{G}(m,r)$. Typically, the faster one can compute $A(U)[H]$, the faster one can minimize $m_U(H)$ in the trust region.

A powerful property of the RTR method is that global convergence of the algorithm toward critical points—local minimizers in practice since it is a descent method—is guaranteed independently of $A(U)$ [ABG07, Thm 4.24, Cor. 4.6]. We take advantage of this and first set it to the identity. This yields a steepest-descent algorithm we later refer to as **RTRMC 1**. Additionally, if we take $A(U)$ to be the Hessian of $f$ at $U$ (16), we get a quadratic convergence rate, even if we only approximately minimize $m_U$ within the trust region using a few steps of a well chosen iterative method [ABG07, Thm 4.14]. This means that the RTR method only requires a few computations of the Hessian along specific directions. We call our method using the Hessian **RTRMC 2**.

## 5  Numerical experiments

We test our algorithms on both synthetic and real data and compare their performances against OptSpace, ADMiRA, SVT, LMaFit and Balanced Factorization in terms of accuracy and computation time. All algorithms are run sequentially by Matlab on the same personal computer[1]. Table 2 specifies a few implementation details.

Table 2: All Matlab implementations call subroutines in non-Matlab code to efficiently deal with the sparsity of the matrices involved. PROPACK [Lar05] is a free package for large and sparse SVD computations.

| Method | Environment | Comment |
|---|---|---|
| RTRMC 1 | Matlab + some C-Mex | Our method with "approximate Hessian" set to identity, i.e., no second order information. $\lambda = 10^{-6}$. For the initial guess $\mathcal{U}_0$, we use the OptSpace trimmed SVD. |
| RTRMC 2 | Matlab + some C-Mex | Same as RTRMC 1 but with exact Hessian. |
| OptSpace | C code | [KO09] with $\lambda = 0$. Trimmed SVD + descent on Grass. |
| ADMiRA | Matlab with PROPACK | [LB10] Matching pursuit based. |
| SVT | Matlab with PROPACK | [CCS08] default $\tau$ and $\delta$. Nuclear norm minimization. |
| LMaFit | Matlab + some C-Mex | [WYZ10] Alternating minimization. |
| Balanced Factorization | Matlab + some C-Mex | [MBS11] One of their Riemannian regression methods. |

Our methods (RTRMC 1 and 2) and Balanced Factorization require knowledge of the target rank $r$. OptSpace, ADMiRA and LMaFit include a mechanism to guess the rank, but benefit from knowing it, hence we provide the true rank to these methods too. As is, the SVT code does not permit the user to specify the rank.

We use the root mean square error (RMSE) criterion to assess the quality of reconstruction of $X$ with $\hat{X}$:

$$\mathrm{RMSE}(X, \hat{X}) = \|X - \hat{X}\|_{\mathrm{F}} / \sqrt{mn}.$$

**Scenario 1.** We first compare convergence behavior of the different methods on synthetic data. We pick $m = n = 10\,000$ and $r = 10$. The dimension of the manifold of $m$-by-$n$ matrices of rank $r$ is $d = r(m+n-r)$. We generate $A \in \mathbb{R}^{m \times r}$ and $B \in \mathbb{R}^{r \times n}$ with i.i.d. normal entries of zero mean and unit variance. The target matrix is $X = AB$. We sample $2.5d$ entries uniformly at random, which yields a sampling ratio of 0.5%. Figure 1 is typical and shows the evolution of the RMSE as a function of time (left) and iteration count (right). For $\hat{X} = UV$ with $U \in \mathbb{R}^{m \times r}$, $V \in \mathbb{R}^{r \times n}$, we compute the RMSE in $\mathcal{O}((m+n)r^2)$ flops using:

$$(mn)\mathrm{RMSE}(AB, UV)^2 = \mathrm{Trace}((A^\top A)(BB^\top)) + \mathrm{Trace}((U^\top U)(VV^\top)) - 2\mathrm{Trace}((U^\top A)(BV^\top)).$$

Be wary though that this formula is numerically inaccurate when the RMSE is much smaller than the norm of either $AB$ or $UV$, owing to the computation of the difference of close large numbers.

**Scenario 2.** In this second test, we repeat the previous experience with rectangular matrices: $m = 1\,000, n = 30\,000, r = 5$ and a sampling ratio of 2.6% ($5d$ known entries). We expect RTRMC to perform well on rectangular matrices since the dimension of the Grassmann manifold we optimize on only grows linearly with $\min(m, n)$, whereas it is the (simple) least-squares problem dimension that grows linearly in $\max(m, n)$. Figure 2 is typical and shows indeed that RTRMC is the fastest tested algorithm on this test.

**Scenario 3.** Following the protocol in [KMO09], we test our method on the Jester dataset 1 [GRGP01] of ratings of a hundred jokes by $24\,983$ users. We randomly select $4\,000$ users and the corresponding continuous ratings in the range $[-10, 10]$. For each user, we extract two ratings at random as test data. We run the different matrix completion algorithms with a prescribed rank on the remaining training data, $N = 100$ times for each rank. Table 3 reports the average Normalized Mean Absolute Error (NMAE) on the test data along with a confidence interval computed as the standard deviation of the NMAE's obtained for the different runs divided by $\sqrt{N}$. All methods but ADMiRA minimize a similar cost function and consequently perform the same.

## 6 Conclusion

Our contribution is an efficient numerical method to solve large low-rank matrix completion problems. RTRMC competes with the state-of-the-art and enjoys proven global and local convergence to local optima, with a quadratic convergence rate for RTRMC 2. Our methods are particularly efficient on rectangular matrices. To obtain such results, we exploited the geometry of the low-rank constraint and applied techniques from the field of optimization on manifolds. Matlab code for RTRMC 1 and 2 is available at:
`http://www.inma.ucl.ac.be/~absil/RTRMC/`.

Table 3: NMAE's on the Jester dataset 1 (Scenario 3). All algorithms solve the problem in well under a minute for rank 7. All but ADMiRA reach similar results. As a reference, consider that a random guesser would obtain a score of 0.33. Goldberg et al. [GRGP01] report a score of 0.187 but use a different protocol.

| rank | RTRMC 2 | OptSpace | LMaFit | Bal. Fac. | ADMiRA |
|---|---|---|---|---|---|
| 1 | $0.1799 \pm 2 \cdot 10^{-4}$ | $0.1799 \pm 2 \cdot 10^{-4}$ | $0.1799 \pm 2 \cdot 10^{-4}$ | $0.1799 \pm 2 \cdot 10^{-4}$ | $0.1836 \pm 2 \cdot 10^{-4}$ |
| 3 | $0.1624 \pm 2 \cdot 10^{-4}$ | $0.1625 \pm 2 \cdot 10^{-4}$ | $0.1624 \pm 2 \cdot 10^{-4}$ | $0.1626 \pm 2 \cdot 10^{-4}$ | $0.1681 \pm 2 \cdot 10^{-4}$ |
| 5 | $0.1584 \pm 2 \cdot 10^{-4}$ | $0.1584 \pm 2 \cdot 10^{-4}$ | $0.1584 \pm 2 \cdot 10^{-4}$ | $0.1584 \pm 2 \cdot 10^{-4}$ | $0.1635 \pm 2 \cdot 10^{-4}$ |
| 7 | $0.1578 \pm 2 \cdot 10^{-4}$ | $0.1581 \pm 2 \cdot 10^{-4}$ | $0.1578 \pm 2 \cdot 10^{-4}$ | $0.1580 \pm 2 \cdot 10^{-4}$ | $0.1618 \pm 2 \cdot 10^{-4}$ |

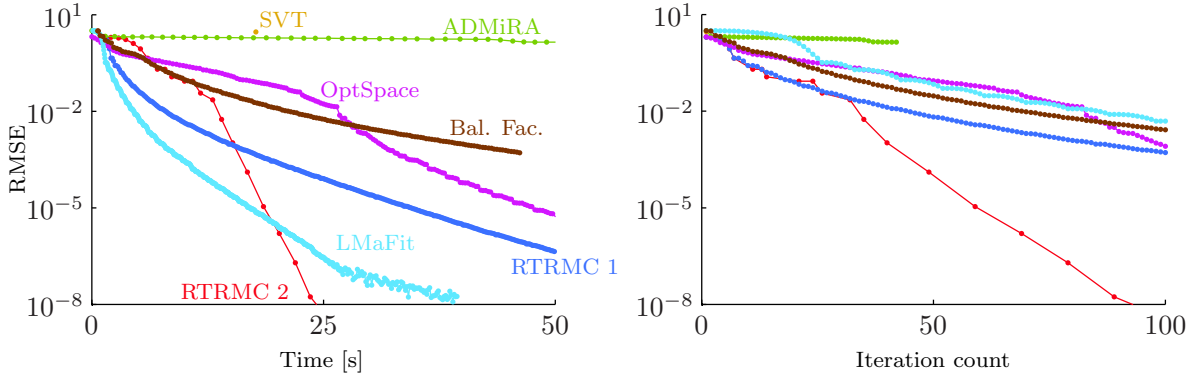

Figure 1: Evolution of the RMSE for the six methods under Scenario 1 ($m = n = 10\,000$, $r = 10$, $|\Omega|/(mn) = 0.5\%$, i.e., 99.5% of the entries are unknown). For RTRMC 2, we count the number of inner iterations, i.e., the number of parallelizable steps. ADMiRA stagnates and SVT diverges. All other methods eventually find the exact solution.

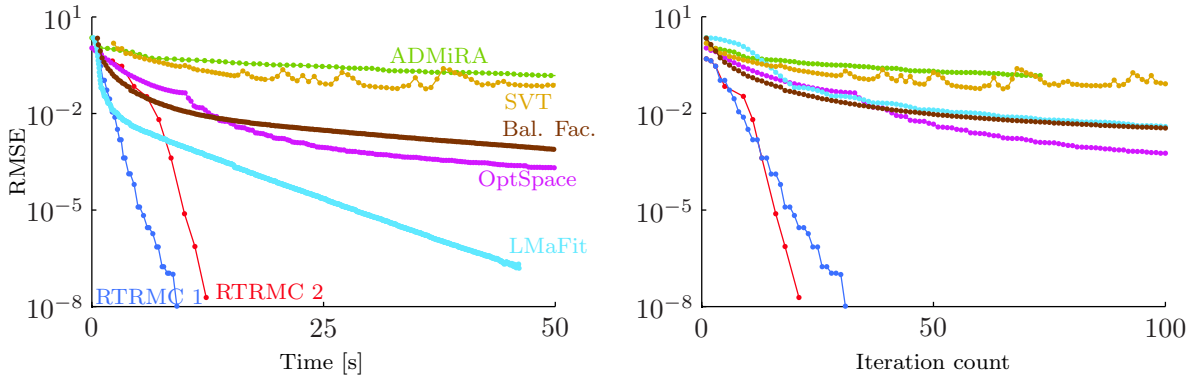

Figure 2: Evolution of the RMSE for the six methods under Scenario 2 ($m = 1\,000$, $n = 30\,000$, $r = 5$, $|\Omega|/(mn) = 2.6\%$). For rectangular matrices, RTRMC is especially efficient owing to the linear growth of the dimension of the search space in $\min(m, n)$, whereas for most methods the growth is linear in $m + n$.

**Acknowledgments**

This paper presents research results of the Belgian Network DYSCO (Dynamical Systems, Control, and Optimization), funded by the Interuniversity Attraction Poles Programme, initiated by the Belgian State, Science Policy Office. NB is an FNRS research fellow (*Aspirant*). The scientific responsibility rests with its authors.

## Footnotes

[1]Intel Core i5 670 @ 3.60GHz (4), 8Go RAM, Matlab 7.10 (R2010a).

# References

[ABG07]   P.-A. Absil, C. G. Baker, and K. A. Gallivan. Trust-region methods on Riemannian manifolds. *Found. Comput. Math.*, 7(3):303–330, July 2007.

[AMS08]   P.-A. Absil, R. Mahony, and R. Sepulchre. *Optimization Algorithms on Matrix Manifolds*. Princeton University Press, Princeton, NJ, 2008.

[BNR10]   L. Balzano, R. Nowak, and B. Recht. Online identification and tracking of subspaces from highly incomplete information. In *Communication, Control, and Computing (Allerton), 2010 48th Annual Allerton Conference on*, pages 704–711. IEEE, 2010.

[Bro05]   M. Brookes. The matrix reference manual. *Imperial College London*, 2005.

[CCS08]   J.F. Cai, E.J. Candès, and Z. Shen. A singular value thresholding algorithm for matrix completion. *Arxiv preprint arXiv:0810.3286*, 2008.

[CR09]   E.J. Candès and B. Recht. Exact matrix completion via convex optimization. *Foundations of Computational Mathematics*, 9(6):717–772, 2009.

[DKM10]   W. Dai, E. Kerman, and O. Milenkovic. A Geometric Approach to Low-Rank Matrix Completion. *Arxiv preprint arXiv:1006.2086*, 2010.

[DMK11]   W. Dai, O. Milenkovic, and E. Kerman. Subspace evolution and transfer (SET) for low-rank matrix completion. *Signal Processing, IEEE Transactions on*, PP(99):1, 2011.

[GRGP01]   K. Goldberg, T. Roeder, D. Gupta, and C. Perkins. Eigentaste: A constant time collaborative filtering algorithm. *Information Retrieval*, 4(2):133–151, 2001.

[KM10]   R.H. Keshavan and A. Montanari. Regularization for matrix completion. In *Information Theory Proceedings (ISIT), 2010 IEEE International Symposium on*, pages 1503–1507. IEEE, 2010.

[KMO09]   R.H. Keshavan, A. Montanari, and S. Oh. Low-rank matrix completion with noisy observations: a quantitative comparison. In *Communication, Control, and Computing, 2009. Allerton 2009. 47th Annual Allerton Conference on*, pages 1216–1222. IEEE, 2009.

[KO09]   R.H. Keshavan and S. Oh. OptSpace: A gradient descent algorithm on the Grassman manifold for matrix completion. *Arxiv preprint arXiv:0910.5260 v2*, 2009.

[Lar05]   R.M. Larsen. PROPACK–Software for large and sparse SVD calculations. *Available online. URL http://sun. stanford. edu/rmunk/PROPACK*, 2005.

[LB10]   K. Lee and Y. Bresler. ADMiRA: Atomic decomposition for minimum rank approximation. *Information Theory, IEEE Transactions on*, 56(9):4402–4416, 2010.

[MBS11]   G. Meyer, S. Bonnabel, and R. Sepulchre. Linear regression under fixed-rank constraints: a Riemannian approach. In *28th International Conference on Machine Learning*. ICML, 2011.

[TB97]   L.N. Trefethen and D. Bau. *Numerical linear algebra*. Society for Industrial Mathematics, 1997.

[Van11]   B. Vandereycken. Low-rank matrix completion by riemannian optimization. Technical report, ANCHP-MATHICSE, Mathematics Section, École Polytechnique Fédérale de Lausanne, 2011.

[WYZ10]   Z. Wen, W. Yin, and Y. Zhang. Solving a low-rank factorization model for matrix completion by a nonlinear successive over-relaxation algorithm. Technical report, Rice University, 2010. CAAM Technical Report TR10-07.

